# Manifold Précis: An Annealing Technique for Diverse Sampling of Manifolds

**Nitesh Shroff** †**, Pavan Turaga** ‡**, Rama Chellappa** †
†Department of Electrical and Computer Engineering, University of Maryland, College Park
‡School of Arts, Media, Engineering and ECEE, Arizona State University
{nshroff,rama}@umiacs.umd.edu, pturaga@asu.edu

## Abstract

In this paper, we consider the Précis problem of sampling $K$ representative yet diverse data points from a large dataset. This problem arises frequently in applications such as video and document summarization, exploratory data analysis, and pre-filtering. We formulate a general theory which encompasses not just traditional techniques devised for vector spaces, but also non-Euclidean manifolds, thereby enabling these techniques to shapes, human activities, textures and many other image and video based datasets. We propose intrinsic manifold measures for measuring the quality of a selection of points with respect to their representative power, and their diversity. We then propose efficient algorithms to optimize the cost function using a novel annealing-based iterative alternation algorithm. The proposed formulation is applicable to manifolds of known geometry as well as to manifolds whose geometry needs to be estimated from samples. Experimental results show the strength and generality of the proposed approach.

## 1 Introduction

The problem of sampling $K$ representative data points from a large dataset arises frequently in various applications. Consider analyzing large datasets of shapes, objects, documents or large video sequences, etc. Analysts spend a large amount of time sifting through the acquired data to familiarize themselves with the content, before using them for their application specific tasks. This has necessitated the problem of optimal selection of a few representative exemplars from the dataset as an important step in exploratory data analysis. Other applications include Internet-based video summarization, where providing a quick overview of a video is important for improving the browsing experience. Similarly, in medical image analysis, picking a subset of $K$ anatomical shapes from a large population helps in identifying the variations within and across shape classes, providing an invaluable tool for analysts.

Depending upon the application, several subset selection criteria have been proposed in the literature. However, there seems to be a consensus in selecting exemplars that are representative of the dataset while minimizing the redundancy between the exemplars. Liu *et al.*[1] proposed that the summary of a document should satisfy the 'coverage' and 'orthogonality' criteria. Shroff *et al.*[2] extended this idea to selecting exemplars from videos that maximize 'coverage' and 'diversity'. Simon *et al.*[3] formulated scene summarization as one of picking interesting and important scenes with minimal redundancy. Similarly, in statistics, stratified sampling techniques sample the population by dividing the dataset into mutually exclusive and exhaustive 'strata' (sub-groups) followed by a random selection of representatives from each strata [4]. The splitting of population into stratas ensures that a diverse selection is obtained. The need to select diverse subsets has also been emphasized in information retrieval applications [5, 6].

Column Subset Selection (CSS) [7, 8, 9] has been one of the popular techniques to address this problem. The goal of CSS is to select the $K$ most "well-conditioned" columns from the matrix of data points. One of the key assumptions behind this and other techniques is that the objects or their representations, lie in the Euclidean space. Unfortunately, this assumption is not valid in many cases. In

applications like computer vision, images and videos are represented by features/models like shapes [10], bags-of-words, linear dynamical systems (LDS) [11], etc. Many of these features/models have been shown to lie in non-Euclidean spaces, implying that the underlying distance metric of the space is not the usual $\ell_2/\ell_p$ norm. Since these feature/model spaces have a non-trivial manifold structure, the distance metrics are highly non-linear functions. Examples of features/models - manifold pairs include: shapes - complex spherical manifold [10], linear subspaces - Grassmann manifold, co-variance matrices - tensor space, histograms - simplex in $\mathbb{R}^n$, etc. Even the familiar bag-of-words representation, used commonly in document analysis, is more naturally considered as a statistical manifold than as a vector space [12]. The geometric properties of the non-Euclidean manifolds allow one to develop accurate inference and classification algorithms [13, 14]. In this paper, we focus on the problem of selecting a subset of $K$ exemplars from a dataset of $N$ points when the dataset has an underlying manifold structure to it. We formulate the notion of representational error and diversity measure of exemplars while utilizing the non-Euclidean structure of the data points followed by the proposal of an efficient annealing-based optimization algorithm.

**Related Work:** The problem of subset selection has been studied by the communities of numerical linear algebra and theoretical computer science. Most work in the former community is related to the *Rank Revealing* QR factorization (RRQR) [7, 15, 16]. Given a data matrix $Y$, the goal of RRQR factorization is to find a permutation matrix $\Pi$ such that the QR factorization of $Y\Pi$ reveals the numerical rank of the matrix. The resultant matrix $Y\Pi$ has as its first $K$ columns the most "well-conditioned" columns of the matrix $Y$. On the other hand, the latter community has focused on Column Subset Selection (CSS). The goal of CSS is to pick $K$ columns forming a matrix $C \in \mathbb{R}^{m \times K}$ such that the residual $\| Y - P_C Y \|_\zeta$ is minimized over all possible choices for the matrix $C$. Here $P_C = CC^\dagger$ denotes the projection onto the $K$-dimensional space spanned by the columns of $C$ and $\zeta$ can represent the spectral or Frobenius norm. $C^\dagger$ indicates the pseudo inverse of matrix $C$. Along these lines, different randomized algorithms have been proposed [17, 18, 9, 8]. Various approaches include a two-stage approach [9], subspace sampling methods [8], etc.

Clustering techniques [19] have also been applied for subset selection [20, 21]. In order to select $K$ exemplars, data points are clustered into $\ell$ clusters with $(\ell \le K)$ followed by the selection of one or multiple exemplars from each cluster to obtain the best representation or low-rank approximation of each cluster. Affinity Propagation [21], is a clustering algorithm that takes similarity measures as input and recursively passes message between nodes until a set of exemplars emerges. As we discuss in this paper, the problems with these approaches are that (a) the objective functions optimized by the clustering functions do not incorporate the diversity of the exemplars, hence can be biased towards denser clusters, and also by outliers, and (b) seeking low-rank approximation of the data matrix or clusters individually is not always an appropriate subset selection criterion. Furthermore, these techniques are largely tuned towards addressing the problem in an Euclidean setting and cannot be applied for datasets in non-Euclidean spaces.

Recently, advances have been made in utilizing non-Euclidean structure for statistical inferences and pattern recognition [13, 14, 22, 23]. These works have addressed inferences, clustering, dimensionality reduction, etc. in non-Euclidean spaces. To the best of our knowledge, the problem of subset selection for analytic manifolds remains largely unaddressed. While one could try to solve the problem by obtaining an embedding of a given manifold into a larger ambient Euclidean space, it is desirable to have a solution that is more intrinsic in nature. This is because the chosen embedding is often arbitrary, and introduces peculiarities that result from such extrinsic approaches. Further manifolds such as the Grassmannian or the manifold of infinite dimensional diffeomorphisms do not admit a natural embedding into a vector space.

**Contributions:** 1) We present the first formal treatment of subset selection for the general case of manifolds, 2) We propose a novel annealing-based alternation algorithm to efficiently solve the optimization problem, 3) We present an extension of the algorithm for data manifolds, and demonstrate the favorable properties of the algorithm on real data.

## 2   Subset Selection on Analytic Manifolds

In this section, we formalize the subset selection problem on manifolds and propose an efficient algorithm. First, we briefly touch upon the necessary basic concepts.

**Geometric Computations on Manifolds:** Let $\mathcal{M}$ be an $m$-dimensional manifold and, for a point $p \in \mathcal{M}$, consider a differentiable curve $\gamma : (-\epsilon, \epsilon) \to \mathcal{M}$ such that $\gamma(0) = p$. The velocity $\dot{\gamma}(0)$

denotes the velocity of $\gamma$ at $p$. This vector is an example of a tangent vector to $\mathcal{M}$ at $p$. The set of all such tangent vectors is called the tangent space to $\mathcal{M}$ at $p$. If $\mathcal{M}$ is a Riemannian manifold then the exponential map $\exp_p : T_p(\mathcal{M}) \to \mathcal{M}$ is defined by $\exp_p(v) = \alpha_v(1)$ where $\alpha_v$ is a specific geodesic. The inverse exponential map (logarithmic map) $\log_p : \mathcal{M} \to T_p$ takes a point on the manifold and returns a point on the tangent space – which is an Euclidean space.

**Representational error on manifolds:** Let us assume that we are given a set of points $X = \{x_1, x_2, \ldots x_n\}$ which belong to a manifold $\mathcal{M}$. The goal is to select a few exemplars $E = \{e_1, \ldots e_K\}$ from the set $X$, such that the exemplars provide a good representation of the given data points, and are minimally redundant. For the special case of vector spaces, two common approaches for measuring representational error is in terms of linear spans, and nearest-exemplar error. The linear span error is given by: $\min_z \|\mathbf{X} - \mathbf{E}z\|_F^2$, where $\mathbf{X}$ is the matrix form of the data, and $\mathbf{E}$ is a matrix of chosen exemplars. The nearest-exemplar error is given by: $\sum_i \sum_{x_k \in \Phi_i} \|x_k - e_i\|^2$, where $e_i$ is the $i^{th}$ exemplar and $\Phi_i$ corresponds to its Voronoi region.

Of these two measures, the notion of linear span, while appropriate for matrix approximation, is not particularly meaningful for general dataset approximation problems since the 'span' of a dataset item does not carry much perceptually meaningful information. For example, the linear span of a vector $x \in \mathbb{R}^n$ is the set of points $\alpha x, \alpha \in \mathbb{R}$. However, if $x$ were an image, the linear span of $x$ would be the set of images obtained by varying the global contrast level. All elements of this set however are perceptually equivalent, and one does not obtain any representational advantage from considering the span of $x$. Further, points sampled from the linear span of few images, would not be meaningful images. This situation is further complicated for manifold-valued data such as shapes, where the notion of linear span does not exist. One could attempt to define the notion of linear spans on the manifold as the set of points lying on the geodesic shot from some fixed pole toward the given dataset item. But, points sampled from this linear span might not be very meaningful e.g., samples from the linear span of a few shapes would give physically meaningless shapes.

Hence, it is but natural to consider the representational error of a set $X$ with respect to a set of exemplars $E$ as follows:

$$J_{rep}(E) = \sum_i \sum_{x_j \in \Phi_i} d_g^2(x_j, e_i) \tag{1}$$

Here, $d_g$ is the geodesic distance on the manifold and $\Phi_i$ is the Voronoi region of the $i^{th}$ exemplar. This boils down to the familiar $K$-means or $K$-medoids cost function for Euclidean spaces. In order to avoid combinatorial optimization involved in solving this problem, we use efficient approximations i.e., we first find the mean followed by the selection of $e_i$ as data point that is closest to the mean. The algorithm for optimizing $J_{rep}$ is given in algorithm 1. Similar to $K$-means clustering, a cluster label is assigned to each $x_j$ followed by the computation of the mean $\mu_i$ for each cluster. This is further followed by selecting representative exemplar $e_i$ as the data point closest to $\mu_i$.

**Diversity measures on manifolds:** The next question we consider is to define the notion of diversity of a selection of points on a manifold. We first begin by examining equivalent constructions for $\mathbb{R}^n$. One of the ways to measure diversity is simply to use the sample variance of the points. This is similar to the construction used recently in [2]. For the case of manifolds, the sample variance can be replaced by the sample Karcher variance, given by the function: $\rho(E) = \frac{1}{K} \sum_{i=1}^{K} d_g^2(\mu, e_i)$, where $\mu$ is the Karcher mean [24], and the function value is the Karcher variance. However, this construction leads to highly inefficient optimization routines, essentially boiling down to a combinatorial search over all possible $K$-sized subsets of $X$.

An alternate formulation for vector spaces that results in highly efficient optimization routines is via Rank-Revealing QR (RRQR) factorizations. For vector spaces, given a set of vectors $\mathbf{X} = \{x_i\}$, written in matrix form $\mathbf{X}$, RRQR [7] aims to find $Q, R$ and a permutation matrix $\Pi \in \mathbb{R}^{n \times n}$ such that $\mathbf{X}\Pi = QR$ reveals the numerical rank of the matrix $\mathbf{X}$. This permutation $\mathbf{X}\Pi = (\mathbf{X}_K \ \mathbf{X}_{n-K})$ gives $\mathbf{X}_K$, the $K$ most linearly independent columns of $\mathbf{X}$. This factorization is achieved by seeking $\Pi$ which maximizes $\Lambda(\mathbf{X}_K) = \prod_i \sigma_i(\mathbf{X}_K)$, the product of the singular values of the matrix $\mathbf{X}_K$.

For the case of manifolds, we adopt an approximate approach in order to measure diversity in terms of the 'well-conditioned' nature of the set of exemplars projected on the tangent space at the mean. In particular, for the dataset $\{x_i\} \subseteq \mathcal{M}$, with intrinsic mean $\mu$, and a given selection of exemplars

**Algorithm 1**: Algorithm to minimize $J_{rep}$

**Input**: $X \in \mathcal{M}$, $k$, index vector $\omega$, $\Gamma$
**Output**: Permutation Matrix $\Pi$
Initialize $\Pi \leftarrow \mathcal{I}_{n \times n}$
**for** $\gamma \leftarrow 1$ **to** $\Gamma$ **do**
    Initialize $\Pi^{(\gamma)} \leftarrow \mathcal{I}_{n \times n}$
    $e_i \leftarrow x_{\omega_i}$ for $i = \{1,2,\dots,k\}$
    **for** $i \leftarrow 1$ **to** $k$ **do**
        $\Phi_i \leftarrow \{x_p : arg\ min_j\ d_g(x_p, e_j) = i\ \}$
        $\mu_i \leftarrow$ mean of $\Phi_i$
        $\hat{j} \leftarrow arg\ min_j\ d_g(x_j, \mu_i)$
        Update: $\Pi^{(\gamma)} \leftarrow \Pi^{(\gamma)}\ \Pi_{i \leftrightarrow \hat{j}}$
    **end**
    Update: $\Pi \leftarrow \Pi\ \Pi^{(\gamma)}$, $\omega \leftarrow \omega \Pi^{(\gamma)}$
    **if** $\Pi^{(\gamma)} = \mathcal{I}_{n \times n}$ **then**
        break
    **end**
**end**

**Algorithm 2**: Algorithm for Diversity Maximization

**Input**: Matrix $V \in \mathbb{R}^{d \times n}$, $k$, Tolerance $tol$
**Output**: Permutation Matrix $\Pi$
Initialize $\Pi \leftarrow \mathcal{I}_{n \times n}$
**repeat**
    Compute $QR$ decomposition of $V$ to obtain
    $R_{11}, R_{12}$ and $R_{22}$ s.t., $V = Q \begin{pmatrix} R_{11} & R_{12} \\ 0 & R_{22} \end{pmatrix}$
    $\beta_{ij} \leftarrow \sqrt{(R_{11}^{-1} R_{12})_{ij}^2 + ||R_{22}\alpha_j||_2^2 ||\alpha_i^T R_{11}^{-1}||_2^2}$
    $\beta_m \leftarrow \max_{ij} \beta_{ij}$
    $(\hat{i}, \hat{j}) \leftarrow arg\ max_{ij} \beta_{ij}$
    Update: $\Pi \leftarrow \Pi\ \Pi_{i \leftrightarrow (j+k)}$
    $V \leftarrow V \Pi_{i \leftrightarrow (j+k)}$
**until** $\beta_m < tol$ ;

$\{e_j\}$, we measure the diversity of exemplars as follows: matrix $\mathbf{T}_E = [\log_\mu(e_j)]$ is obtained by projecting the exemplars $\{e_j\}$ on the tangent space at mean $\mu$. Here, $\log()$ is the inverse exponential map on the manifold and gives tangent vectors at $\mu$ that point towards $e_j$.

Diversity can then be quantified as $J_{div}(E) = \Lambda(\mathbf{T}_E)$, where, $\Lambda(\mathbf{T}_E)$ represents the product of the singular values of the matrix $\mathbf{T}_E$. For vector spaces, this measure is related to the sample variance of the chosen exemplars. For manifolds, this measure is related to the sample Karcher *variance*. If we denote $\mathbf{T}_X = [\log_\mu(x_i)]$, the matrix of tangent vectors corresponding to all data-points, and if $\Pi$ is the permutation matrix that orders the columns such that the first $K$ columns of $\mathbf{T}_X$ correspond to the most diverse selection, then

$$J_{div}(E) = \Lambda(\mathbf{T}_E) = det(R_{11}), \text{where, } \mathbf{T}_X \Pi = QR = Q \begin{pmatrix} R_{11} & R_{12} \\ 0 & R_{22} \end{pmatrix} \quad (2)$$

Here, $R_{11} \in \mathbb{R}^{K \times K}$ is the upper triangular matrix of $R \in \mathbb{R}^{n \times n}$, $R_{12} \in \mathbb{R}^{K \times (n-K)}$ and $R_{22} \in \mathbb{R}^{(n-K) \times (n-K)}$. The advantage of viewing the required quantity as the determinant of a sub-matrix on the right hand-side of the above equation is that one can obtain efficient techniques for optimizing this cost function. The algorithm for optimizing $J_{div}$ is adopted from [7] and described in algorithm 2. Input to the algorithm is a matrix $V$ created by the tangent-space projection of $X$ and output is the $K$ most "well-conditioned" columns of $V$. This is achieved by first decomposing $V$ into $QR$ and computing $\beta_{ij}$, which indicates the benefit of swapping $i^{th}$ and $j^{th}$ columns [7]. The algorithm then selects pair $(\hat{i}, \hat{j})$ corresponding to the maximum benefit swap $\beta_m$ and if $\beta_m > tol$, this swap is accepted. This is repeated until either $\beta_m < tol$ or maximum number of iterations is completed.

**Algorithm 3**: Annealing-based Alternation Algorithm for Subset Selection on Manifolds

**Input**: Data points $X = \{x_1, x_2, \dots, x_n\} \in \mathcal{M}$, Number of exemplars $k$, Tolerance step $\delta$
**Output**: $E = \{e_1, \dots e_k\} \subseteq X$
**Initial setup:**
Compute intrinsic mean $\mu$ of $X$
Compute tangent vectors $v_i \leftarrow \log_\mu(x_i)$
$V \leftarrow [v_1, v_2, \dots, v_n]$
$\omega \leftarrow [1, 2, \dots, n]$ be the $1 \times n$ index vector of $X$
$Tol \leftarrow 1$
Initialize: $\Pi \leftarrow$ Randomly permute columns of $\mathcal{I}_{n \times n}$
Update: $V \leftarrow V\Pi$, $\omega \leftarrow \omega \Pi$.
**while** $\Pi \neq \mathcal{I}_{n \times n}$ **do**
    **Diversity:** $\Pi \leftarrow \text{Div}(V, k, tol)$ as in algorithm 2.
    Update: $V \leftarrow V\Pi$, $\omega \leftarrow \omega \Pi$.
    **Representative Error:** $\Pi \leftarrow \text{Rep}(X, k, \omega, 1)$ as in algorithm 1
    Update: $V \leftarrow V\Pi$, $\omega \leftarrow \omega \Pi$.
    $tol \leftarrow tol + \delta$
**end**
$e_i \leftarrow x_{\omega_i}$ for $i = \{1,2,\dots,k\}$

**Representation and Diversity Trade-offs for Subset Selection:** From (1) and (2), it can be seen that we seek a solution that represents a trade-off between two conflicting criteria. As an example, in figure 1(a) we show two cases, where $J_{rep}$ and $J_{div}$ are individually optimized. We can see that the solutions look quite different in each case. One way to write the global cost function is as a weighted combination of the two. However, such a formulation does not lend itself to efficient optimization routines (c.f. [2]). Further, the choice of weights is often left unjustified. Instead, we propose an annealing-based alternating technique of optimizing the conflicting criteria $J_{rep}$

| Symbol | Represents |
|---|---|
| $\Gamma$ | Maximum number of iterations |
| $\mathcal{I}_{n \times n}$ | Identity matrix |
| $\Phi_i$ | Voronoi region of $i^{th}$ exemplar |
| $\Pi_{i \mapsto j}$ | Permutation matrix that swaps columns $i$ and $j$ |
| $\Pi^{(\gamma)}$ | $\Pi$ in the $\gamma^{th}$ iteration |
| $V$ | Matrix obtained by tangent-space projection of $X$ |
| $H_{ij}$ | $(i, j)$ element of matrix $H$ |
| $\alpha_j$ | $j^{th}$ column of the identity matrix |
| $H\alpha_j, \alpha_j^T H$ | $j^{th}$ column and row of matrix $H$ respectively |

Table 1: Notations used in Algorithm 1 - 3

| Computational Step | Complexity |
|---|---|
| $\mathcal{M}$ Exponential Map (assume) | $O(\nu)$ |
| $\mathcal{M}$ Inverse exponential Map (assume) | $O(\chi)$ |
| Intrinsic mean of $X$ | $O((n\chi + \nu)\Gamma)$ |
| Projection of $X$ to tangent-space | $O(n\chi)$ |
| Geodesic distances in alg. 1 | $O(nK\chi)$ |
| $K$ intrinsic means | $O((n\chi + K\nu)\Gamma)$ |
| Alg. 2 | $O(mnK \log n)$ |
| $\mathcal{G}_{m,p}$ Exponential Map | $O(p^3)$ |
| $\mathcal{G}_{m,p}$ Inverse exponential map | $O(p^3)$ |

Table 2: Complexity of various computational steps.

and $J_{div}$. Optimization algorithms for $J_{rep}$ and $J_{div}$ individually are given in algorithms 1 and 2 respectively. We first optimize $J_{div}$ to obtain an initial set of exemplars, and use this set as an initialization for optimizing $J_{rep}$. The output of this stage is used as the current solution to further optimize $J_{div}$. However, with each iteration, we increase the tolerance parameter $tol$ in algorithm 2. This has the effect of accepting only those permutations that increase the diversity by a higher factor as iterations progress. This is done to ensure that the algorithm is guided towards convergence. If the $tol$ value is not increased at each iteration, then optimizing $J_{div}$ will continue to provide a new solution at each iteration that modifies the cost function only marginally. This is illustrated in figure 1(c), where we show how the cost functions $J_{rep}$ and $J_{div}$ exhibit an oscillatory behavior if annealing is not used. As seen in figure 1(b), the convergence of $J_{div}$ and $J_{rep}$ is obtained very quickly on using the proposed annealing alternation technique. The complete annealing based alternation algorithm is described in algorithm 3. A technical detail to be noted here is that for algorithm 2, input matrix $V \in \mathbb{R}^{d \times n}$ should have $d \geq k$. For cases where $d < k$, algorithm 2 can be replaced by its extension proposed in [9]. Table 1 shows the notations introduced in algorithms 1 - 3. $\Pi_{i \mapsto j}$ is obtained by permuting $i$ and $j$ columns of the identity matrix.

## 3 Complexity, Special cases and Limitations

In this section, we discuss how the proposed method relates to the special case of $\mathcal{M} = \mathbb{R}^n$, and to sub-manifolds of $\mathbb{R}^n$ specified by a large number of samples. For the case of $\mathbb{R}^n$, the cost functions $J_{rep}$ and $J_{div}$ boil down to familiar notions of clustering and low-rank matrix approximation respectively. In this case, algorithm 3 reduces to alternation between clustering and matrix approximation with the annealing ensuring that the algorithm converges. This results in a new algorithm for subset-selection in vector spaces.

For the case of manifolds implicitly specified using samples, one can approach the problem in one of two ways. The first would be to obtain an embedding of the space into a Euclidean space and apply the special case of the algorithm for $\mathcal{M} = \mathbb{R}^n$. The embedding here needs to preserve the geodesic distances between all pairs of points. Multi-dimensional scaling can be used for this purpose. However, recent methods have also focused on estimating logarithmic maps numerically from sampled data points [25]. This would make the algorithm directly applicable for such cases, without the need for a separate embedding. Thus the proposed formalism can accommodate manifolds with known and unknown geometries.

However, the formalism is limited to manifolds of finite dimension. The case of infinite dimensional manifolds, such as diffeomorphisms [26], space of closed curves [27], etc. pose problems in formulating the diversity cost function. While $J_{div}$ could have been framed purely in terms of pairwise geodesics, making it extensible to infinite dimensional manifolds, it would have made the optimization a significant bottleneck, as already discussed in section 2.

**Computational Complexity:** The computational complexity of computing exponential map and its inverse is specific to each manifold. Let $n$ be the number of data points and $K$ be the number of exemplars to be selected. Table 2 enumerates the complexity of different computational step of the algorithm. The last two rows show the complexity of an efficient algorithm proposed by [28] to compute the exponential map and its inverse for the case of Grassmann manifold $\mathcal{G}_{m,p}$.

## 4 Experiments

**Baselines:** We compare the proposed algorithm with two baselines. The first baseline is a clustering-based solution to subset selection, where we cluster the dataset into $K$ clusters, and pick as exemplar the data point that is closest to the cluster centroid. Since clustering optimizes only the

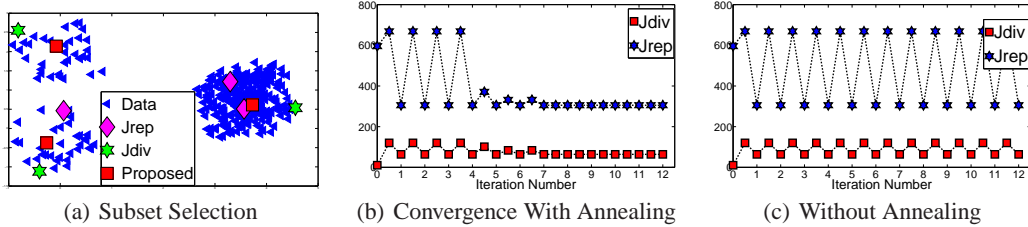

|        (a) Subset Selection        |   (b) Convergence With Annealing   |        (c) Without Annealing       |

Figure 1: Subset selection for a simple dataset consisting of unbalanced classes in $\mathbb{R}^4$. (a) Data projected on $\mathbb{R}^2$ for visualization using PCA. While trying to minimize the representational error, $J_{rep}$ picks two exemplars from the dominant class. $J_{div}$ picks diverse exemplars but from the boundaries. The proposed approach strikes a balance between the two and picks one 'representative' exemplar from each class. Convergence Analysis of algorithm 3: (b) with annealing and (c) without annealing.

representation cost-function, we do not expect it to have the diversity of the proposed algorithm. This corresponds to the special case of optimizing only $J_{rep}$. The second baseline is to apply a tangent-space approximation to the entire data-set at the mean of the dataset, and then apply a subset-selection algorithm such as RRQR. This corresponds to optimizing only $J_{div}$ where the input matrix is the matrix of tangent vectors. Since minimization of $J_{rep}$ is not explicitly enforced, we do not expect the exemplars to be the best representatives, even though the set is diverse.

**A Simple Dataset:** To gain some intuition, we first perform experiments on a simple synthetic dataset. For easy visualization and understanding, we generated a dataset with 3 unbalanced classes in Euclidean space $\mathbb{R}^4$. Individual cost functions, $J_{rep}$ and $J_{div}$ were first optimized to pick three exemplars using algorithms 1 and 2 respectively. Selected exemplars have been shown in figure 1(a), where the four dimensional dataset has been projected into two dimensions for visualization using Principal Component Analysis (PCA). Despite unbalanced class sizes, when optimized individually, $J_{div}$ seeks to select exemplars from diverse classes but tends to pick them from the class boundaries. While unbalanced class sizes cause $J_{rep}$ to pick 2 exemplars from the dominant cluster. Algorithm 3 iteratively optimizes for both these cost functions and picks an exemplar from every class. These exemplars, are closer to the centroid of the individual classes.

Figure 1(b) shows the convergence of the algorithm for this simple dataset and compares it with the case when no annealing is applied (figure 1(c)). $J_{rep}$ and $J_{div}$ plots are shown as the iterations of algorithm 3 progresses. When annealing is applied, the tolerance value ($tol$) is increased by $0.05$ in each iteration. It can be noted that in this case the algorithm converges to a steady state in 7 iterations ($tol = 1.35$). If no annealing is applied, the algorithm does not converge.

**Shape sampling/summarization:** We conducted a real shape summarization experiment on the MPEG dataset [29]. This dataset has 70 shape classes with 20 shapes per class. For our experiments, we created a smaller dataset of 10 shape classes with 10 shapes per class. Figure 2(a) shows the shapes used in our experiments. We use an affine-invariant representation of shapes based on landmarks. Shape boundaries are uniformly sampled to obtain $m$ landmark points. These points are concatenated in a matrix to obtain the landmark matrix $\mathcal{L} \in \mathbb{R}^{m \times 2}$. Left singular vectors ($U_{m \times 2}$), obtained by the singular value decomposition of matrix $\mathcal{L} = U\Sigma V^T$, give the affine-invariant representation of shapes [30]. This affine shape-space $U$ of $m$ landmark points is a 2-dimensional subspace of $\mathbb{R}^m$. These $p$-dimensional subspaces in $\mathbb{R}^m$ constitute the Grassmann manifold $\mathcal{G}_{m,p}$. Details of the algorithms for the computation of exponential and inverse exponential map on $\mathcal{G}_{m,p}$ can be found in [28] and has also been included in the supplemental material.

In the experiment, the cardinality of the subset was set to 10. As the number of shape classes is also 10, one would ideally seek one exemplar from each class. Algorithms 1 and 2 were first individually optimized to select the optimal subset. Algorithm 1 was applied intrinsically on the manifold with multiple initializations. Figure 2(b) shows the output with the least cost among these initializations. For algorithm 2, data points were projected on the tangent space at the mean using the inverse exponential map and the selected subset is shown in figure 2(c). Individual optimization of $J_{rep}$ results in 1 exemplar each from 6 classes, 2 each from 2 classes ('apple' and 'flower') and misses 2 classes ('bell' and 'chopper'). While, individual optimization of $J_{div}$ alone picks 1 each from 8 classes, 2 from the class 'car' and none from the class 'bell'. It can be observed that exemplars chosen by $J_{div}$ for classes 'glass', 'heart','flower' and 'apple' tend to be unusual members of the

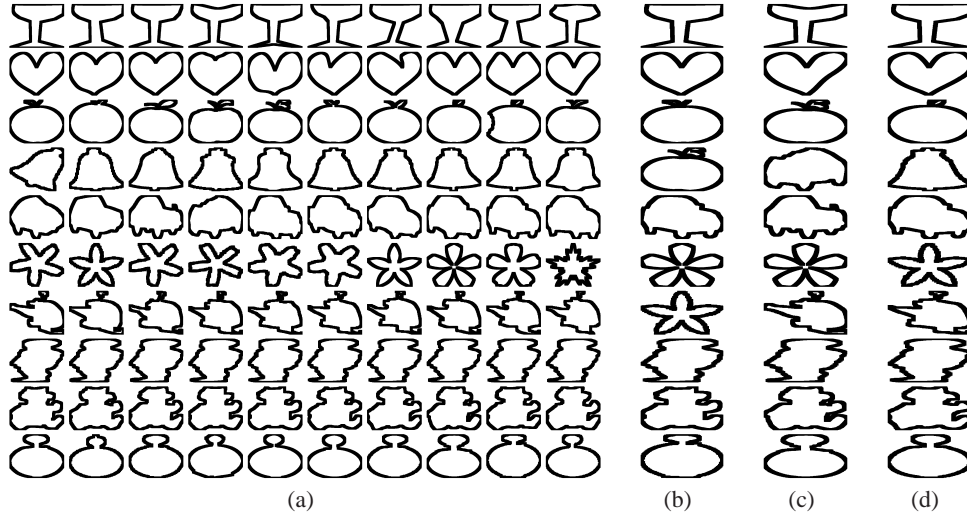

|  | (a) |  |  |  |  |  |  |  | (b) | (c) | (d) |

Figure 2: (a) 10 classes from MPEG dataset with 10 shapes per class. Comparison of 10 exemplars selected by (b)$J_{rep}$, (c) $J_{div}$ and (d) Proposed Approach. $J_{rep}$ picks 2 exemplars each from 2 classes ('apple' and 'flower') and misses 'bell' and 'chopper' classes. $J_{div}$ picks 1 from 8 different classes, 2 exemplars from class 'car' and none from class 'bell'. It can be observed that exemplars chosen by $J_{div}$ for classes 'glass', 'heart', 'flower' and 'apple' tend to be unusual members of the class. It also picks up the flipped car. While the proposed approach picks one representative exemplars from each class as desired.

class. It also picks up the flipped car. Optimizing for both $J_{div}$ and $J_{rep}$ using algorithm 3 picks one 'representative' exemplar from each class as shown in figure 2(d).

These exemplars picked by the three algorithms can be further used to label data points. Table 3 shows the confusion table thus obtained. For each data point, we find the nearest exemplar, and label the data point with the ground-truth label of this exemplar. For example, consider the row labeled as 'bell'. All the data points of the class 'bell' were labeled as 'pocket' by $J_{rep}$ while $J_{div}$ labeled 7 data points from this class as 'chopper' and 3 as 'pocket'. This confusion is largely due to both $J_{rep}$ and $J_{div}$ having missed out picking exemplars from this class. The proposed approach correctly labels all data points as it picks exemplars from every class.

|  | Glass | Heart | Apple | Bell | Baby | Chopper | Flower | Car | Pocket | Teddy |
|---|---|---|---|---|---|---|---|---|---|---|
| Glass | (10,10,10) |  |  |  |  |  |  |  |  |  |
| Heart |  | (10,10,10) |  |  |  |  |  |  |  |  |
| Apple |  | (0,1,0) | (8,7,10) |  |  | (2,0,0) |  |  | (0,2,0) |  |
| Bell |  |  |  | (0,0,10) |  | (0,7,0) |  |  | (10,3,0) |  |
| Baby |  |  |  |  | (10,10,10) |  |  |  |  |  |
| Chopper |  | (2,0,0) | (8,0,0) |  |  | (0,10,10) |  |  |  |  |
| Flower |  |  |  |  |  |  | (10,10,10) |  |  |  |
| Car |  |  |  |  |  |  |  | (10,10,10) |  |  |
| Pocket |  |  |  |  |  |  |  |  | (10,10,10) |  |
| Teddy |  |  |  |  |  |  |  |  |  | (10,10,10) |

Table 3: Confusion Table. Entries correspond to the tuple $(J_{rep}, J_{div}, Proposed)$. Row labels correspond to the ground truth labels of the shape and the column labels correspond to the label of the nearest exemplar. Only non-zero entries have been shown in the table.

**KTH human action dataset:** The next experiment was conducted on the KTH human action dataset [31]. This dataset consists of videos with 6 actions conducted by 25 persons in 4 different scenarios. For our experiment, we created a smaller dataset of 30 videos with the first 5 human subjects conducting 6 actions in the $s4$ (indoor) scenario. Figure 3(a) shows sample frames from each video. This dataset mainly consists of videos captured under constrained settings. This makes it difficult to identify the 'usual' or 'unusual' members of a class. To better understand the performance of the three algorithms, we synthetically added occlusion to the last video of each class. These occluded videos serve as the 'unusual' members.

Histogram of Oriented Optical Flow (HOOF) [32] was extracted from each frame to obtain a normalized time-series for the videos. A Linear Dynamical System (LDS) is then estimated from this time-series using the approach in [11]. This model is described by the state transition equation: $x(t + 1) = Ax(t) + w(t)$ and the observation equation $z(t) = Cx(t) + v(t)$, where

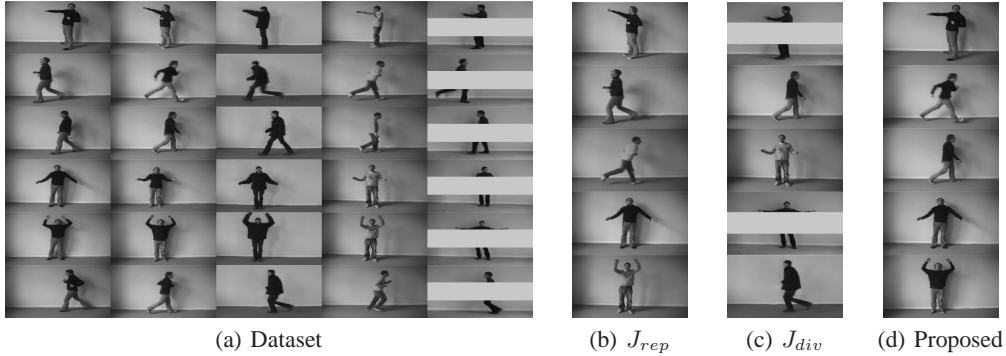

|  (a) Dataset | (b) $J_{rep}$ | (c) $J_{div}$ | (d) Proposed |

Figure 3: (a) Sample frames from KTH action dataset [31]. From top to bottom action classes are { box, run, walk, hand-clap, hand-wave and jog }. 5 exemplars selected by: (b)$J_{rep}$, (c) $J_{div}$ and (d) Proposed. Exemplars picked by $J_{rep}$ correspond to { box, run, run, hand-clap, hand-wave } actions. While $J_{div}$ selects { box, walk, hand-clap, hand-wave and jog }. Proposed approach picks { box, run, walk, hand-clap and hand-wave }.

$x \in \mathbb{R}^d$ is the hidden state vector, $z \in \mathbb{R}^p$ is the observation vector, $w(t) \sim N(0, \Theta)$ and $v(t) \sim N(0, \Xi)$ are the noise components. Here, $A$ is the state-transition matrix and $C$ is the observation matrix. The expected observation sequence of model $(A, C)$ lies in the column space of the infinite extended 'observability' matrix which is commonly approximated by a finite matrix $O_m^T = [C^T, (CA)^T, (CA^2)^T, \ldots, (CA^{m-1})^T]$. The column space of this matrix $O_m^T \in \mathbb{R}^{mp \times d}$ is a $d$-dimensional subspace and hence lies on the Grassmann manifold.

In this experiment, we consider the scenario when the number of classes in a dataset is unknown. We asked the algorithm to pick 5 exemplars when the actual number of classes in the dataset is 6. Figure 3(b) shows one frame from each of the videos selected when $J_{rep}$ was optimized alone. It picks 1 exemplar each from 3 classes ('box','hand-clap' and 'hand-wave'), 2 from the class 'run' while misses out on 'walk' and 'jog'. On the other hand, $J_{div}$ (when optimized alone) picks 1 each from 5 different classes and misses the class 'run'. It can be seen that $J_{div}$ picks 2 exemplars that are 'unusual' members (occluded videos) of their respective class. The proposed approach picks 1 representative exemplar from 5 classes and none from the class 'jog'. The proposed approach achieves both a diverse selection of exemplars, and also avoids picking outlying exemplars.

**Effect of Parameters and Initialization:** In our experiments, the effect of tolerance steps ($\delta$) for smaller values ($< 0.1$) has very minimal effect. After a few attempts, we fixed this value to $0.05$ for all our experiments. In the first iteration, we start with $tol = 1$. With this value, algorithm 2 accepts any swap that increases $J_{div}$. This makes output of algorithm 2 after first iteration almost insensitive to initialization. While, in the later iterations, swaps are accepted only if they increase the value of $J_{div}$ significantly and hence input to algorithm 2 becomes more important with the increase in $tol$.

## 5 Conclusion and Discussion

In this paper, we addressed the problem of selecting $K$ exemplars from a dataset when the dataset has an underlying manifold structure to it. We utilized the geometric structure of the manifold to formulate the notion of picking exemplars which minimize the representational error while maximizing the diversity of exemplars. An iterative alternation optimization technique based on annealing has been proposed. We discussed its convergence and complexity and showed its extension to data manifolds and Euclidean spaces. We showed summarization experiments with real shape and human actions dataset. Future work includes formulating subset selection for infinite dimensional manifolds and efficient approximations for this case. Also, several special cases of the proposed approach point to new directions of research such as the cases of vector spaces and data manifolds.

**Acknowledgement:** This research was funded (in part) by a grant $N00014 - 09 - 1 - 0044$ from the Office of Naval Research. The first author would like to thank Dikpal Reddy and Sima Taheri for helpful discussions and their valuable comments.

## References

[1] K. Liu, E. Terzi, and T. Grandison, "ManyAspects: a system for highlighting diverse concepts in documents," in *Proceedings of VLDB Endowment*, 2008.

[2] N. Shroff, P. Turaga, and R. Chellappa, "Video Précis: Highlighting diverse aspects of videos," *IEEE Transactions on Multimedia*, vol. 12, no. 8, pp. 853 –868, Dec. 2010.

[3] I. Simon, N. Snavely, and S. Seitz, "Scene summarization for online image collections," in *ICCV*, 2007.

[4] W. Cochran, *Sampling techniques*. Wiley, 1977.

[5] Y. Yue and T. Joachims, "Predicting diverse subsets using structural svms," in *ICML*, 2008.

[6] J. Carbonell and J. Goldstein, "The use of mmr, diversity-based reranking for reordering documents and reproducing summaries," in *SIGIR*, 1998.

[7] M. Gu and S. Eisenstat, "Efficient algorithms for computing a strong rank-revealing QR factorization," *SIAM Journal on Scientific Computing*, vol. 17, no. 4, pp. 848–869, 1996.

[8] P. Drineas, M. Mahoney, and S. Muthukrishnan, "Relative-error CUR matrix decompositions," *SIAM Journal on Matrix Analysis and Applications*, vol. 30, pp. 844–881, 2008.

[9] C. Boutsidis, M. Mahoney, and P. Drineas, "An improved approximation algorithm for the column subset selection problem," in *SODA*, 2009.

[10] D. Kendall, "Shape manifolds, Procrustean metrics and complex projective spaces," *Bulletin of London Mathematical society*, vol. 16, pp. 81–121, 1984.

[11] S. Soatto, G. Doretto, and Y. N. Wu, "Dynamic textures," *ICCV*, 2001.

[12] J. D. Lafferty and G. Lebanon, "Diffusion kernels on statistical manifolds," *Journal of Machine Learning Research*, vol. 6, pp. 129–163, 2005.

[13] P. T. Fletcher, C. Lu, S. M. Pizer, and S. C. Joshi, "Principal geodesic analysis for the study of nonlinear statistics of shape," *IEEE Transactions on Medical Imaging*, vol. 23, no. 8, pp. 995–1005, August 2004.

[14] A. Srivastava, S. H. Joshi, W. Mio, and X. Liu, "Statistical shape analysis: Clustering, learning, and testing," *IEEE Transactions on pattern analysis and machine intelligence*, vol. 27, no. 4, 2005.

[15] G. Golub, "Numerical methods for solving linear least squares problems," *Numerische Mathematik*, vol. 7, no. 3, pp. 206–216, 1965.

[16] T. Chan, "Rank revealing QR factorizations," *Linear Algebra and Its Applications*, vol. 88, pp. 67–82, 1987.

[17] A. Frieze, R. Kannan, and S. Vempala, "Fast Monte-Carlo algorithms for finding low-rank approximations," *Journal of the ACM (JACM)*, vol. 51, no. 6, pp. 1025–1041, 2004.

[18] A. Deshpande and L. Rademacher, "Efficient volume sampling for row/column subset selection," in *Foundations of Computer Science (FOCS)*, 2010.

[19] G. Gan, C. Ma, and J. Wu, *Data clustering: theory, algorithms, and applications*. Society for Industrial and Applied Mathematics, 2007.

[20] I. Dhillon and D. Modha, "Concept decompositions for large sparse text data using clustering," *Machine learning*, vol. 42, no. 1, pp. 143–175, 2001.

[21] B. J. Frey and D. Dueck, "Clustering by passing messages between data points," *Science*, vol. 315, pp. 972–976, Feb. 2007.

[22] R. Subbarao and P. Meer, "Nonlinear mean shift for clustering over analytic manifolds," in *CVPR*, 2006.

[23] A. Goh and R. Vidal, "Clustering and dimensionality reduction on riemannian manifolds," in *CVPR*, 2008.

[24] H. Karcher, "Riemannian center of mass and mollifier smoothing," *Communications on Pure and Applied Mathematics*, vol. 30, no. 5, pp. 509–541, 1977.

[25] T. Lin and H. Zha, "Riemannian manifold learning," *IEEE Transactions on Pattern Analysis and Machine Intelligence*, vol. 30, pp. 796–809, 2008.

[26] A. Trouvé, "Diffeomorphisms groups and pattern matching in image analysis," *International Journal of Computer Vision*, vol. 28, pp. 213–221, July 1998.

[27] W. Mio, A. Srivastava, and S. Joshi, "On shape of plane elastic curves," *International Journal of Computer Vision*, vol. 73, no. 3, pp. 307–324, 2007.

[28] K. Gallivan, A. Srivastava, X. Liu, and P. Van Dooren, "Efficient algorithms for inferences on grassmann manifolds," in *IEEE Workshop on Statistical Signal Processing*, 2003.

[29] L. Latecki, R. Lakamper, and T. Eckhardt, "Shape descriptors for non-rigid shapes with a single closed contour," in *CVPR*, 2000.

[30] E. Begelfor and M. Werman, "Affine invariance revisited," in *CVPR*, 2006.

[31] C. Schuldt, I. Laptev, and B. Caputo, "Recognizing human actions: a local SVM approach," in *ICPR*, 2004.

[32] R. Chaudhry, A. Ravichandran, G. Hager, and R. Vidal, "Histograms of oriented optical flow and binet-cauchy kernels on nonlinear dynamical systems for the recognition of human actions," in *CVPR*, 2009.

